# Spatio-Temporal Interactive Learning for Efficient Image Reconstruction of Spiking Cameras

**Bin Fan**[1*]    **Jiaoyang Yin**[2,3*]    **Yuchao Dai**[4]    **Chao Xu**[1]    **Tiejun Huang**[2,3]    **Boxin Shi**[2,3†]

[1]Nat'l Key Lab of General AI, School of Intelligence Science and Technology, Peking University
[2]State Key Lab of Multimedia Info. Processing, School of Computer Science, Peking University
[3]Nat'l Eng. Research Ctr. of Visual Technology, School of Computer Science, Peking University
[4]School of Electronics and Information, Northwestern Polytechnical University
`{binfan,shiboxin,tjhuang}@pku.edu.cn`, `yinjiaoyang@stu.pku.edu.cn`,
`xuchao@cis.pku.edu.cn`, `daiyuchao@nwpu.edu.cn`

## Abstract

The spiking camera is an emerging neuromorphic vision sensor that records high-speed motion scenes by asynchronously firing continuous binary spike streams. Prevailing image reconstruction methods, generating intermediate frames from these spike streams, often rely on complex step-by-step network architectures that overlook the intrinsic collaboration of spatio-temporal complementary information. In this paper, we propose an efficient spatio-temporal interactive reconstruction network to jointly perform inter-frame feature alignment and intra-frame feature filtering in a coarse-to-fine manner. Specifically, it starts by extracting hierarchical features from a concise hybrid spike representation, then refines the motion fields and target frames scale-by-scale, ultimately obtaining a full-resolution output. Meanwhile, we introduce a symmetric interactive attention block and a multi-motion field estimation block to further enhance the interaction capability of the overall network. Experiments on synthetic and real-captured data show that our approach exhibits excellent performance while maintaining low model complexity. The code is available at `https://github.com/GitCVfb/STIR`.

## 1 Introduction

High-speed imaging has become a high-profile topic in fields such as autonomous driving, industrial monitoring, and robotics, due to its ability to precisely capture the continuous light intensity behaviour in a scene. Conventional digital cameras often rely on expensive specialized sensors when capturing fast-moving objects, so the trade-off between frame rate and cost has limited the widespread adoption and further development of high-speed cameras. In recent years, neuromorphic cameras, especially event cameras [31, 40, 1, 19] and spiking cameras [7, 26], have emerged as innovative vision sensors. They possess characteristics such as high temporal resolution, high dynamic range, and low latency, opening up new possibilities for high-speed imaging of consumer-grade cameras.

The spiking camera achieves integral sampling with 40,000Hz by emulating the central fovea's sampling mechanism in the retina [34, 47]. Each photoreceptive unit continuously and independently captures photons, and asynchronously fires spikes once the accumulated intensity exceeds a given threshold. Unlike event cameras that only record relative changes in light intensity (*i.e.*, differential sampling), spiking cameras have the ability to encode the absolute light intensity because the spike firing rate is proportional to the scene brightness. Consequently, the spiking camera can preserve

---

[*]Equal contribution.
[†]Corresponding author.

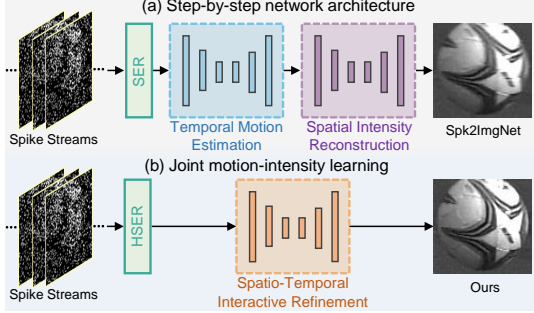

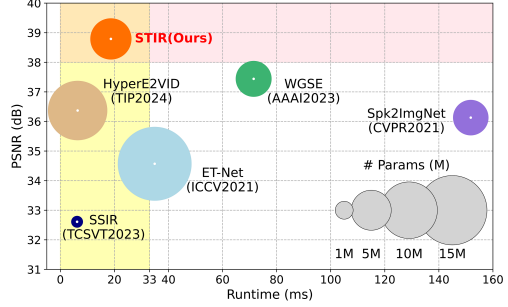

Figure 1: Different paradigms of spike-to-image reconstruction. (a) Prevailing step-by-step network architecture (*e.g.*, Spk2ImgNet [55]). (b) Our proposed joint motion-intensity learning framework. A simple yet effective hybrid spike embedding representation (HSER) is also proposed as a link between the binary spikes and the deep model.

Figure 2: Model comparison of PSNR, runtime, and model size. The PSNR is calculated on the SREDS dataset [57]. The runtime is tested using an RTX 3090 GPU on real-captured data [66] with a spatial resolution of $400 \times 250$. Our model achieves favorable results in terms of accuracy and efficiency.

more sufficient scene texture information, making it highly promising for pixel-level tasks, such as image reconstruction [65, 55, 3, 8], depth estimation [53, 46], semantic segmentation [52, 64], and optical flow estimation [23, 59, 49]. However, spiking cameras solely record dense binary time-sequence information, making it difficult to directly apply existing vision algorithms designed for conventional frame-based cameras. To reconstruct dynamic scene content from asynchronous spike streams, traditional methods either exploit the temporal statistical characteristics [65], *e.g.*, texture from playback (TFP) and texture from inter-spike-intervals (TFI), or mimic the human physiological mechanisms, *e.g.*, retina-like visual imaging [66] and short-term plasticity [63, 62]. Nonetheless, noise and motion blur frequently present a tricky trade-off throughout the dynamic scene reconstruction process, which could lead to less than ideal reconstruction results. In contrast, deep learning-based methods [55, 4, 57], with their powerful representation capabilities to mine latent spatio-temporal cues from spike streams through end-to-end learning, offer a more promising way to address the dynamic scene reconstruction problem of spiking cameras.

Deep learning-based methods usually cascade three independent modules: spike embedding representation (SER), temporal motion estimation, and spatial intensity recovery, as illustrated in Fig. 1 (a). The first module [56, 52, 60, 59, 58, 49] typically extracts time-series information from the spike stream, serving as an essential bridge between the binary spikes and the deep model. The temporal motion estimation module either explicitly estimates the motion field [56, 9] or implicitly establishes temporal motion correlations (*e.g.*, deformable convolution [55, 60], attention [4, 5]), aiming to align context in the feature space. Following this, an additional spatial intensity recovery module [56, 4, 60] is added to reconstruct the intermediate frame from the aligned feature representations. Although this design paradigm of first estimating motion and then reconstructing images has achieved reasonably good results, it hinders the information interaction and joint optimization in time and space, creating a bottleneck for further improving the image reconstruction quality of spiking cameras. On the one hand, motion estimation and intensity recovery are inherently a "chicken-and-egg" problem: more accurate motion modeling will lead to better intermediate frame reconstruction, and vice versa. On the other hand, this step-by-step combination tends to reduce inference efficiency, which is detrimental to efficient deployment in real-world applications.

In this paper, we point out that temporal motion estimation and spatial intensity recovery can be mutually reinforcing, as shown in Fig. 1 (b). To this end, we design an efficient **S**patio-**T**emporal **I**nteractive **R**econstruction network, termed **STIR**. Specifically, we first deliver a concise hybrid spike embedding representation (HSER) into a hierarchical feature encoder to obtain pyramid features at different granularities. Then, a spatio-temporal interactive decoder is proposed to enable the joint refinement of spatio-temporal complementary information from coarse to fine. In particular, inter-frame feature alignment and intra-frame feature filtering can be performed simultaneously. The former mainly focuses on temporal motion cues to complete warping-based feature registration, while the latter progressively maintains purer image features through synthesis. In addition, we integrate a symmetric interactive attention block at the top-level pyramid and introduce a multi-motion

field estimation block at the bottom-level pyramid, further upgrading the network's spatio-temporal interaction ability. To fully harness the network's potential, a simple yet effective HSER module is also devised, which incorporates the common advantages of explicit spike representation based on internal statistics (with better certainty and explainability) and implicit spike representation based on neural networks (with stronger expressive power). Extensive experimental results on synthetic and real-captured data demonstrate that our approach significantly outperforms state-of-the-art (SOTA) image reconstruction methods, with a 1.35dB improvement in PSNR while also enjoying fast inference speed, as shown in Fig. 2.

The main contributions of this paper can be summarized as follows:

1) We propose STIR, an efficient and flexible framework for image reconstruction of spiking cameras, which facilitates joint learning of complementary motion and intensity information.
2) We design a symmetric interactive attention block that enhances the bilateral correlation between the intermediate frame and temporal contextual features.
3) We develop a simple yet effective hybrid spike embedding representation module with both good interpretability and strong expressive power.

## 2 Related Works

**Neuromorphic Cameras.** Neuromorphic cameras mimic neurobiological structures and functionalities of the retina. Different from conventional frame-based cameras, they operate asynchronously at the pixel level, allowing each pixel to act independently. Two main types of neuromorphic cameras include event cameras, *e.g*., DVS [31], DAVIS [37], ATIS [40], CeleX [31], and spiking cameras [7, 26]. Event cameras utilize a differential sampling approach, triggering events only when changes in illuminance surpass a specific logarithmic threshold. Conversely, spiking cameras follow an integral sampling method, where photon accumulation leads to spike firing once a given threshold is reached. Therefore, event cameras produce sparser outputs, while spiking cameras provide a more regular input format for reconstructing absolute light intensity.

**Event-to-image Reconstruction.** Deep learning-based methods for reconstructing intensity images from events have demonstrated significant progress. E2VID [41] is a seminal work for this purpose by using a recurrent fully convolutional network. Following E2VID, numerous studies have augmented it from various angles, including FireNet [42], E2VID++ [43], FireNet++ [43]. Also, SPADE layers and Transformer were integrated into E2VID in [2, 48], which enhanced the quality but at a higher cost. HyperE2VID [10] used hypernetworks to generate per-pixel adaptive filters and adopted a dynamic neural network architecture. However, since event cameras solely record changes in relative light intensity, they struggle to reconstruct the texture details of visual scenes.

**Spike-to-image Reconstruction.** In the task of spike-to-image reconstruction, traditional methods usually leverage the temporal statistical properties of spiking cameras. Zhu *et al*. [65] explored the spike generation principle and proposed two basic methods, *a.k.a*., TFP and TFI. Zhao *et al*. [56] hierarchically merged short- and long-term filtering. Another line of work focuses on mimicking human physiological mechanisms [66, 65, 63, 62]. Deep learning techniques have also propelled advancements in this challenging task. Spk2ImgNet [55] was the first CNN-based architecture and achieved impressive results. The wavelet transform was combined with CNN-based learnable modules in [52]. Recently, an energy-efficient scheme was developed [57] based on the spiking neural network (SNN). High-dynamic-range and high-frame-rate videos were generated in [3] by introducing the rolling readout mechanism [13, 11, 16, 14, 15, 17, 18, 12]. Furthermore, several self-supervised CNNs [4, 5] have also been developed to alleviate the dependence on synthetic datasets. However, due to the step-by-step paradigm, the above CNN-based architectures inevitably have higher model complexity, blocking them from mobile and real-time applications. In contrast, our STIR model jointly considers temporal motion estimation and spatial intensity recovery, thus facilitating the intrinsic collaboration of spatio-temporal complementary information.

## 3 Preliminaries

### 3.1 Working Mechanism of the Spiking Camera

The spiking camera employs an "integrate-and-fire" mechanism. Each pixel independently and continuously receives photons from the scene and converts them into photoelectrons, which are then

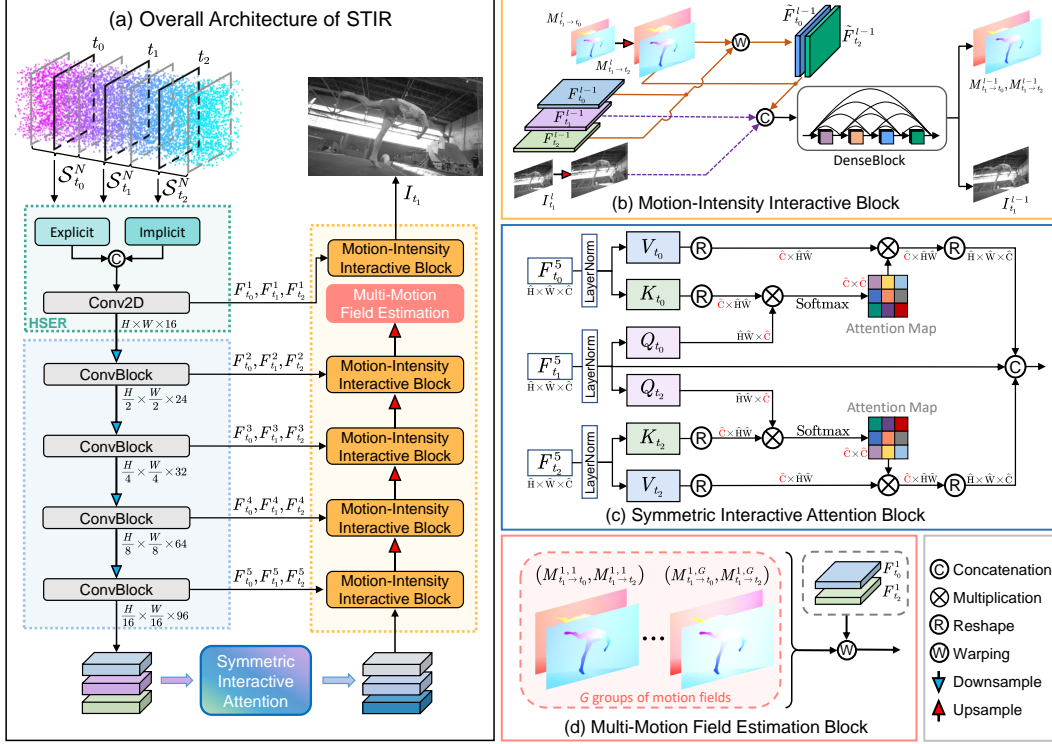

Figure 3: Overview of our STIR framework (a) and details of the key components (b), (c), and (d). In our joint learning architecture, spatio-temporal features are refined progressively from coarse to fine, where warping-based inter-frame feature alignment (Orange line) and synthesis-based intra-frame feature filtering (Purple dashed line) are simultaneously performed in (b). Integrating (c) and (d) at the top and bottom pyramids, respectively, can boost the spatio-temporal interaction of the network.

accumulated via an integrator. When the accumulated photoelectrons exceed the predetermined threshold, the spiking camera asynchronously fires a spike, while clearing the pixel's photoelectrons to start a new accumulation cycle. The working mechanism can be formulated as [56, 26, 61]:

$$\mathbf{A}_t(\mathbf{x}) = \int_0^t \alpha I_\tau(\mathbf{x}) \mathrm{d}\tau \mod \theta, \tag{1}$$

where $\mathbf{A}_t(\mathbf{x})$ reflects the number of photoelectrons accumulated at pixel $\mathbf{x} = (x, y)$ in the integrator. $I_\tau(\mathbf{x})$ represents the light intensity at pixel $\mathbf{x}$ at timestamp $\tau$. The photoelectric conversion rate is denoted as $\alpha$ and the firing threshold is set to $\theta$. Assume that $\delta$ (in microsecond level) is used to quantify a spike accumulation cycle, the spiking camera can output a dense binary spike plane with a spatial resolution of $H \times W$ at timestamp $n\delta, n \in \mathbb{N}$. As a result, during an "integrate-and-fire" process with a temporal length of $N$, the spatio-temporal resolution of the spike stream $\mathcal{S}_t^N$ will reach $H \times W \times N$, where $t$ denotes the central timestamp of $\mathcal{S}_t^N$.

## 3.2  Problem Statement

Given a continuous binary spike stream $\mathcal{S}_{t_1}^{3N} \in \{0, 1\}^{H \times W \times 3N}$ with a spatio-temporal resolution of $H \times W \times 3N$, centered at timestamp $t_1$, similar to [4, 5], we divide it evenly into three non-overlapping spike sub-streams $\mathcal{S}_{t_0}^N$, $\mathcal{S}_{t_1}^N$, and $\mathcal{S}_{t_2}^N$ in chronological order, centered at timestamps $t_0$, $t_1$, and $t_2$, respectively. This paper aims to reconstruct an intermediate intensity frame $I_{t_1}$ corresponding to timestamp $t_1$. Notably, under such a problem setting, in order to recover $I_{t_1}$ successfully, $\mathcal{S}_{t_1}^N$ can be utilized to model the intrinsic representation of spatial features corresponding to $t_1$, while $\mathcal{S}_{t_0}^N$ and $\mathcal{S}_{t_2}^N$ can be exploited to complement the contextual information in the temporal domain.

# 4 Methodology

## 4.1 Overview

The overall network architecture is depicted in Fig. 3. We first propose a hybrid spike embedding representation module in Sec. 4.2, which characterizes the three spike sub-streams $\mathcal{S}_{t_0}^N$, $\mathcal{S}_{t_1}^N$, and $\mathcal{S}_{t_2}^N$ as feature maps $F_{t_0}$, $F_{t_1}$, and $F_{t_2}$ corresponding to timestamps $t_0$, $t_1$, and $t_2$, respectively. Subsequently, to enable spatio-temporal interactions at more granularity, they are adopted to produce multi-scale pyramid features, including intermediate features $\{F_{t_1}^l\}_{l=1}^L$ and temporal contextual features $\{F_{t_0}^l\}_{l=1}^L$, $\{F_{t_2}^l\}_{l=1}^L$, through a weight-sharing hierarchical feature encoder in Sec. 4.3. Here, $L$ indicates the number of pyramid levels. Then, a compact symmetric interactive attention block is leveraged in Sec. 4.4 to initially model the bilateral correlations between the top-level pyramid features $\mathcal{S}_{t_1}^N$ and $\{\mathcal{S}_{t_0}^N, \mathcal{S}_{t_2}^N\}$. Finally, warping-based inter-frame feature alignment and synthesis-based intra-frame feature filtering are jointly executed across the spatio-temporal interactive decoder in Sec. 4.5. Therefore, progressive motion-intensity collaboration is achieved in a single encoder-decoder. Additionally, we estimate $G$ groups of motion fields at the bottom-level pyramid, which helps to improve the performance and robustness of the whole model.

## 4.2 Hybrid Spike Embedding Representation

The spike embedding representation is dedicated to mining time-series information from the input spike stream, serving as a crucial link between the spike stream and the deep network model. To encode the corresponding light intensity features from the spike stream, a simple strategy is to utilize *explicit* spike representation approaches based on internal statistics, such as TFP [65] or TFI [65]. This approach builds on the temporal statistical characteristics of spiking cameras, which can physically provide good interpretability and relatively stable intensity frames for spike-to-image reconstruction tasks like video frame interpolation tasks [30, 29, 45]. Nevertheless, this strategy often struggles to balance noise and motion blur, resulting in limited feature expression capabilities. Another more effective way is to *implicitly* engineer more robust features via CNNs [52, 55, 5, 4, 49, 58, 60]. However, due to the lack of certainty, the spike embedding features obtained in this manner will change when the network parameters are updated during training, limiting the efficient alignment of context in the temporal motion estimation process. In summary, we hope to seek a tractable spike embedding representation method that not only offers good certainty and strong expressive capability but also maintains low computational cost.

To this end, we propose HSER to combine the advantages of explicit and implicit spike representations. Specifically, for the input spike sub-stream $\mathcal{S}_{t_i}^N, i = \{0, 1, 2\}$, we first obtain multiple explicit spike representations using the widely-used TFP method [65] based on varying temporal windows. This is inspired by [55, 4], because short windows give better details but bring noise, while long windows can suppress noise but easily introduce blur. At the same time, we also feed $\mathcal{S}_{t_i}^N$ into a ResNet [21] block for implicit modeling. Finally, these resulting features are concatenated along the channel dimension, and then the spike embedding feature $F_{t_i}$ is generated through a 2D convolution.

## 4.3 Hierarchical Feature Encoder

After obtaining the spike embedding features $F_{t_0}$, $F_{t_1}$, and $F_{t_2}$ corresponding to the three continuous spike sub-streams, we set them as the bottom-level features $F_{t_0}^1$, $F_{t_1}^1$, and $F_{t_2}^1$ of the feature pyramid. On this basis, a hierarchical feature encoder is designed to build an $L$-level feature pyramid, such that multi-granularity feature representations $\{F_{t_0}^l\}_{l=1}^L$, $\{F_{t_1}^l\}_{l=1}^L$, and $\{F_{t_2}^l\}_{l=1}^L$ are extracted from the spike streams. Note that the network parameters are shared across the three spike sub-streams. At each level of the pyramid, we use a $3 \times 3$ 2D convolution with a stride of 2 for feature downsampling, followed by a residual block [21]. Additionally, a PReLU activation [20] is appended after each 2D convolution. The number of feature channels at the $l$-level pyramid is $C_l$. In the following, $\{F_{t_0}^l\}_{l=1}^L$, $\{F_{t_1}^l\}_{l=1}^L$, and $\{F_{t_2}^l\}_{l=1}^L$ will facilitate inter- and intra-frame interactive learning from coarse to fine.

## 4.4 Symmetric Interactive Attention

Recently, the transformer has demonstrated its capability to model long-range correlations between features in various visual tasks [32, 53, 33, 39, 48]. To inject prior motion-intensity guidance

into the subsequent interactive decoder, we present an effective and efficient symmetric interactive attention block. It leverages the multi-head cross-attention mechanism at the top-level pyramid to symmetrically capture the mutual dependencies between $F_{t_1}^L$ and $\{F_{t_0}^L, F_{t_2}^L\}$. Especially, we utilize the intermediate feature $F_{t_1}^L$ as the *query*, while making temporal contextual features $F_{t_0}^L$ and $F_{t_2}^L$ as *key/value* separately, to ensure symmetric interaction with the *query*.

As illustrated in Fig. 3 (c), we linearly project each component (*i.e.*, *query*, *key*, and *value*) by applying layer normalization. In this way, the intermediate feature $F_{t_1}^L$ is projected into two *queries*, $Q_{t_0}$ and $Q_{t_2}$, respectively. At the same time, the temporal contextual features $F_{t_0}^L$ and $F_{t_2}^L$ are projected into two *keys*, $K_{t_0}$ and $K_{t_2}$, as well as two *values*, $V_{t_0}$ and $V_{t_2}$, respectively. The attention-based bilateral correlations can be symmetrically computed as follows:

$$\text{Attention}_{t_1 \to t_i} = \text{Softmax}\left(\frac{K_{t_i}^T Q_{t_i}}{\alpha_i}\right) V_{t_i}, \quad i = 0, 2, \tag{2}$$

where $\alpha_i$ denotes the learnable scaling parameter used to control the magnitude of the dot product. Similar to [51, 6, 50], we perform multi-head *query-key* feature interaction along the channel rather than spatial dimensions, which can effectively enhance computational efficiency due to linear complexity instead of quadratic. By aggregating local and non-local contexts, the final interaction feature $\chi^L \in \mathbb{R}^{H/2^{L-1} \times W/2^{L-1} \times 3C_L}$ can be yielded as:

$$\chi^L = \text{Conv}\left(\left[\text{Attention}_{t_1 \to t_0}, F_{t_1}^L, \text{Attention}_{t_1 \to t_2}\right]\right), \tag{3}$$

where $[\,]$ denotes a channel-wise concatenation operation and Conv is a point-wise convolution layer Note that the intrinsic intermediate feature $F_{t_1}^L$ is preserved through skip connections.

## 4.5 Spatio-Temporal Interactive Decoder

Instead of using a step-by-step network architecture like [56, 4, 5, 60], we propose to jointly and progressively perform temporal motion estimation and spatial intensity recovery (*cf.*, Fig. 1 (b)), thereby maximizing their complementary advantages across spatio-temporal contextual features. Specifically, we develop an efficient motion-intensity interactive block to simultaneously accomplish warping-based inter-frame feature alignment and synthesis-based intra-frame feature filtering, which is inspired by well-established event-based video frame interpolation methods [45, 44, 27]. Notably, warping can integrate light intensity information over the time series, while synthesis can mitigate the influence of spike fluctuations. By cascading multiple motion-intensity interactive blocks from coarse to fine granularity, the intermediate frame can be progressively decoded. Moreover, at the bottom-level pyramid, we propose to predict multiple motion fields to aggregate more comprehensive temporal contexts, which is beneficial to further improve image reconstruction quality.

**Motion-Intensity Interactive Block.** The network details are depicted in Fig. 3 (a) and (b). At the top-level pyramid, $\chi^L$ is input to the motion-intensity interactive block, which simultaneously estimates the motion fields $M_{t_1 \to t_0}^L, M_{t_1 \to t_2}^L$ and synthesizes the intermediate intensity frame $I_{t_1}^L$ using a dense block [25]. Subsequently, at the $L-1$ level of the pyramid, $M_{t_1 \to t_0}^L$ and $M_{t_1 \to t_2}^L$ are upsampled to backward warp the temporal contextual features $F_{t_0}^{L-1}, F_{t_2}^{L-1}$, thereby registering the spatio-temporal information around timestamps $t_0$ and $t_2$ to the intermediate timestamp $t_1$. This process is referred to as *inter-frame feature alignment*, where feature warping is expressed as:

$$\tilde{F}_{t_i}^{L-1} = \mathcal{W}\left(F_{t_i}^{L-1}, \uparrow M_{t_1 \to t_i}^L\right), \quad i = 0, 2, \tag{4}$$

where $\uparrow$ indicates the upsampled variables, $\tilde{F}_{t_i}^{L-1}$ represents the warped feature candidate at the $L-1$ level. Meanwhile, we bilinearly upsample $I_{t_1}^L$ and then concatenate it with the corresponding intermediate feature $F_{t_1}^{L-1}$, followed by a 2D convolution to reduce the influence of spike fluctuations. This process essentially implements *intra-frame feature filtering* through the synthesis of the upsampled frame $\uparrow I_{t_1}^L$ and the intermediate feature $F_{t_1}^{L-1}$. Formally,

$$S_{t_1}^{L-1} = \text{Conv2D}\left(\left[F_{t_1}^{L-1}, \uparrow I_{t_1}^L\right]\right). \tag{5}$$

Note that motion-based warping is more effective in handling significant pixel displacements but is less robust to occlusions. Conversely, intensity-based synthesis exhibits better robustness to occlusions and inconsistent brightness but may degrade image quality in short-time spike sub-streams. To this

Table 1: Quantitative comparisons against SOTA methods on the synthetic SREDS dataset [57] and real-captured dataset [66]. Best and second-best results are **boldfaced** and underlined, respectively. Thanks to the spatio-temporal interaction, our approach consistently demonstrates optimal reconstruction performance, along with excellent parameter size, GPU memory usage, and FLOPs.

| Method | Params (M) | Memory (G) | FLOPs (T) | Synthetic Dataset PSNR↑ | SSIM↑ | LPIPS↓ | NIQE↓ | BRISQUE↓ | Real Dataset NIQE↓ | BRISQUE↓ |
|---|---|---|---|---|---|---|---|---|---|---|
| TFP [65] | – | – | – | 25.35 | 0.690 | 0.2547 | 5.970 | 43.074 | 9.342 | 45.202 |
| TFI [65] | – | – | – | 18.50 | 0.638 | 0.2590 | 4.518 | 44.933 | 10.09 | 58.309 |
| TFSTP [63] | – | – | – | 20.68 | 0.618 | 0.2761 | 5.348 | 51.697 | 10.92 | 64.566 |
| SSIR [57] | 0.38 | 10.4 | **0.24** | 32.61 | 0.919 | 0.0500 | 3.467 | 15.664 | 5.750 | 25.341 |
| ET-Net [48] | 16.7 | 17.7 | 0.52 | 34.57 | 0.938 | 0.0535 | 3.400 | 17.155 | 6.512 | 17.393 |
| HyperE2VID [10] | 10.7 | **6.87** | 0.43 | 36.37 | 0.947 | 0.0506 | 3.126 | 16.774 | 6.306 | 17.020 |
| Spk2ImgNet [55] | 3.76 | 14.6 | 9.17 | 36.13 | 0.950 | 0.0294 | 3.084 | 15.348 | 5.662 | 16.518 |
| WGSE [52] | 3.85 | 19.7 | 3.93 | 37.44 | 0.958 | 0.0241 | 3.032 | 15.555 | 5.620 | 16.154 |
| STIR (Ours) | 5.11 | 9.20 | 0.42 | **38.79** | **0.966** | **0.0183** | **2.915** | **14.835** | **5.394** | **15.854** |

end, we further purchase a dense block to merge the complementary advantages of warping-based and synthesis-based features, *i.e.*,

$$I_{t_1}^{L-1}, M_{t_1 \to t_0}^{L-1}, M_{t_1 \to t_2}^{L-1} = \text{DenseBlock}\left(\left[S_{t_1}^{L-1}, \tilde{F}_{t_0}^{L-1}, \tilde{F}_{t_2}^{L-1}, \uparrow M_{t_1 \to t_0}^{L}, \uparrow M_{t_1 \to t_2}^{L}\right]\right). \quad (6)$$

Therefore, information sharing and mutual collaboration of spatio-temporal features can be achieved by progressively refining the motion-intensity interactive blocks from $L$-level to 1-level pyramids.

**Multi-Motion Field Estimation Block.** Multi-motion field estimation has been proven to be a feasible strategy to improve reconstruction quality in video frame interpolation tasks [30, 24, 29]. Inspired by this, we simply amplify the output channels in the bottom-level pyramid to estimate $G$ groups of motion fields $\left\{M_{t_1 \to t_0}^{1,g}, M_{t_1 \to t_2}^{1,g} \mid g \in [1, G]\right\}$. Hence, $G$ groups of warped feature candidates can be appended with diversity at full resolution, as shown in Fig. 3 (d). This is beneficial for compensating additional details when local inaccuracies occur in a single group of motion fields, thereby enhancing spatio-temporal interaction capabilities. The analyses are detailed in Sec. 5.3.

### 4.6 Loss Function

We employ the combination of reconstruction loss $\mathcal{L}_{\text{rec}}$, perceptual loss $\mathcal{L}_{\text{per}}$, and multi-scale consistency loss $\mathcal{L}_{\text{msc}}$ as the total loss function $\mathcal{L}$ to train our network, namely,

$$\mathcal{L} = \mathcal{L}_{\text{rec}} + \lambda_{\text{per}}\mathcal{L}_{\text{per}} + \mathcal{L}_{\text{msc}}, \quad (7)$$

where we empirically set $\lambda_{\text{per}}$ to 0.2. The $\ell_1$ distance between the final predicted image and the ground truth image is measured in $\mathcal{L}_{\text{rec}}$, *i.e.*,

$$\mathcal{L}_{\text{rec}} = \frac{1}{HW}\left\|\hat{I}_{t_1}^1 - I_{t_1}^1\right\|_1. \quad (8)$$

We also introduce $\mathcal{L}_{\text{per}}$ to mitigate the blurry effect and preserve more details, that is,

$$\mathcal{L}_{\text{per}} = \frac{1}{HW}\|\phi_{\text{vgg}}(\hat{I}_{t_1}^1) - \phi_{\text{vgg}}(I_{t_1}^1)\|_2, \quad (9)$$

where $\phi_{\text{vgg}}$ is the feature extractor of the pre-trained VGG-Net. Furthermore, we propose $\mathcal{L}_{\text{msc}}$ to force the multi-scale intermediate intensity frames $\{I_{t_1}^l\}_{l=2}^L$, synthesized from the 2-level to $L$-level pyramids, to be consistent with the ground truth. The $\ell_1$ distance can be formulated as follows:

$$\mathcal{L}_{\text{msc}} = \frac{1}{HW(L-1)}\sum_{l=2}^{L}\frac{1}{2^{l-1}}\left\|\hat{I}_{t_1}^l - I_{t_1}^l\right\|_1. \quad (10)$$

## 5 Experiments

### 5.1 Implementation Details

**Datasets.** We adopt the recently released SREDS dataset [57], which is synthesized based on the REDS dataset [38], for network training. It is divided into 240 training scenes and 30 testing scenes.

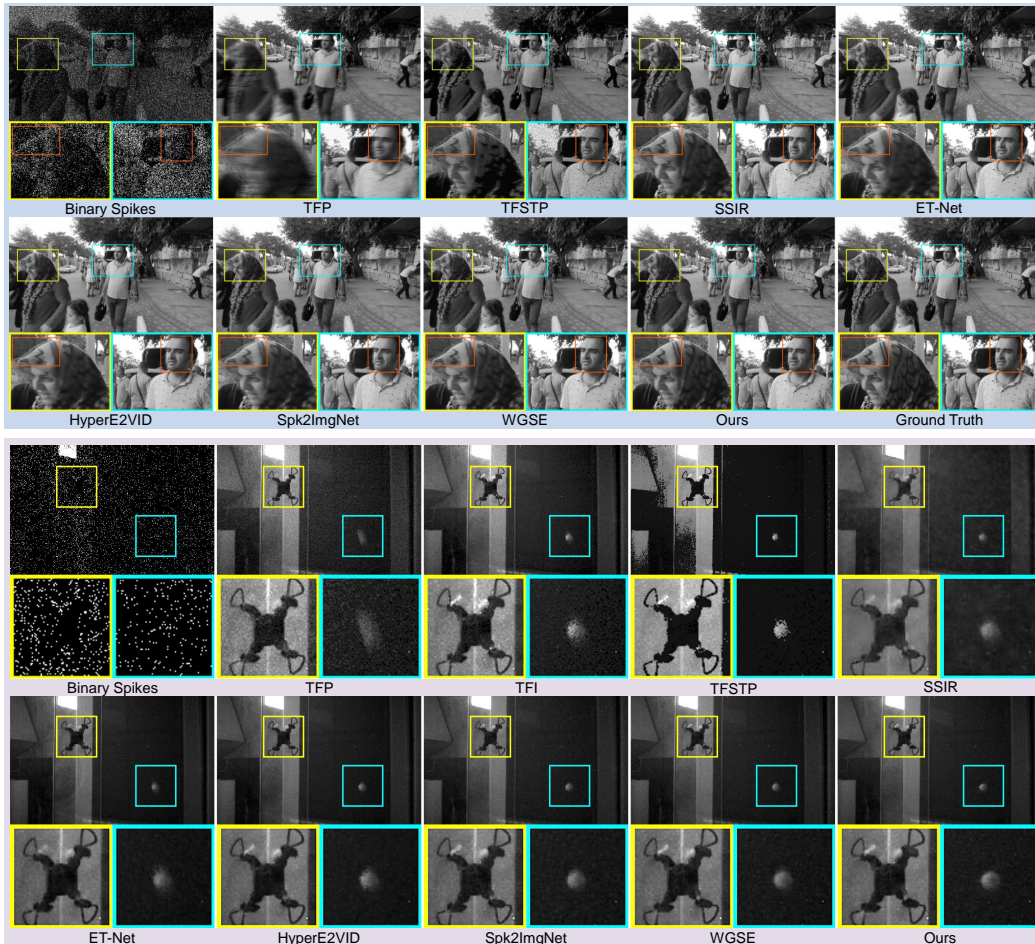

Figure 4: Visual comparison on synthetic [57] (top) and real-captured [61] (bottom) data. Our method reconstructs precise boundaries of fast-moving objects with higher fidelity. Zoom in for more details.

Each scene contains 24 consecutive frames, with a corresponding spike stream of $N = 20$ centered around each frame. The spatial resolution is $1280 \times 720$. During training, each scene is cropped non-overlappingly to $96 \times 96$, yielding a total of 21,840 patches. We evaluate our model using real-captured data, including: 1) The publicly available "momVidarReal2021" [61] and "recVidarReal2019" [66] datasets (with $400 \times 250$ resolution, containing high-speed motion of objects and cameras, and also used in [62, 52]). 2) Real spike data ($1000 \times 1000$) collected by ourselves using a spiking camera.

**Training Details.** Our model is trained using the Adam optimizer [28] for 150 epochs with a batch size of 8. The initial learning rate is 0.0001 and decays by a factor of 0.7 every 50 epochs. The temporal length of the input spike stream is 60, *i.e.*, $N = 20$. The number of pyramid levels is set to 5, *i.e.*, $L = 5$. Note that the reconstruction loss $\mathcal{L}_{\text{rec}}$, perceptual loss $\mathcal{L}_{\text{per}}$, and multi-scale consistency loss $\mathcal{L}_{\text{msc}}$ are used together to train our network. In our HSER module, we construct a 5-channel TFP-based explicit representation with a scaling step of 4, as well as an 11-channel ResNet-based implicit representation, for each spike sub-stream. Thus, the number of feature channels is 16, 24, 32, 64, and 96, respectively. Besides, 3 groups of motion fields are estimated at the bottom-level pyramid, *i.e.*, $G = 3$. Spikes and ground truth images are randomly flipped vertically as well as rotated $90°$, $180°$, or $270°$ during training. All models are trained and tested on a single NVIDIA RTX 3090 GPU.

**Evaluation Metrics.** We apply standard PSNR and SSIM metrics and learned perceptual metric LPIPS [54] to measure the visual quality quantitatively. Moreover, two non-reference image quality assessment metrics NIQE [36] and BRISQUE [35] are employed. A higher PSNR/SSIM (↑) or lower LPIPS/NIQE/BRISQUE (↓) score indicates better performance.

**Comparison Methods.** We compare our method with the following four types of baselines. 1) **Traditional methods**: TFP [65], TFI [65], and TFSTP [63]. 2) **SNN-based**: SSIR [57], designed

Table 3: **Ablation studies on the SREDS dataset [57]**. Underlining indicates our full model.

(a) **Feature pyramid levels.** The more granular hierarchical features can promote superior results due to finer feature alignment and refinement.

|  | PSNR | SSIM | #Paras | TFLOPs |
|---|---|---|---|---|
| 3-level | 37.99 | 0.960 | **0.832** | **0.377** |
| 4-level | 38.02 | 0.962 | 2.119 | 0.403 |
| 5-level | **38.79** | **0.966** | 5.107 | 0.419 |

(b) **Motion-intensity interaction.** Combining warping-based and synthesis-based features for coarse-to-fine refinement can significantly improve reconstruction quality.

|  | PSNR | SSIM | LPIPS | #Paras | TFLOPs |
|---|---|---|---|---|---|
| w/o warping | 37.74 | 0.960 | 0.023 | 4.872 | 0.263 |
| w/o synthesis | 12.19 | 0.414 | 0.685 | 5.088 | 0.416 |
| Full model | **38.79** | **0.966** | **0.018** | 5.107 | 0.419 |

(c) **Symmetric interactive attention.** Removing it or replacing it with independent cross-attention mechanisms both result in lower reconstruction accuracy.

|  | PSNR | SSIM | #Paras | TFLOPs |
|---|---|---|---|---|
| Removing | 38.09 | 0.962 | 4.967 | 0.4186 |
| Independent | 38.23 | 0.963 | 5.318 | 0.4218 |
| Interactive | **38.79** | **0.966** | 5.107 | 0.4194 |

(d) **Multi-motion field estimation.** We investigate different groups of motion fields. More motion field favours compensation for additional image details.

|  | PSNR | SSIM | LPIPS | #Paras | TFLOPs |
|---|---|---|---|---|---|
| 1 | 37.89 | 0.961 | 0.0211 | 4.944 | 0.273 |
| 3 | **38.79** | **0.966** | **0.0183** | 5.107 | 0.419 |
| 5 | 38.37 | 0.964 | 0.0207 | 5.366 | 0.654 |

(e) **Model capacity.** $\times N$ denotes the width multiplier for the feature channel. Our method offers good flexibility and the performance is better with larger model capacity.

|  | PSNR | SSIM | #Paras | TFLOPs |
|---|---|---|---|---|
| $\times 0.5$ | 38.09 | 0.962 | **1.554** | **0.325** |
| $\times 1.0$ | 38.79 | 0.966 | 5.107 | 0.419 |
| $\times 1.5$ | 38.80 | **0.967** | 10.94 | 0.568 |
| $\times 2.0$ | **38.86** | **0.967** | 19.06 | 0.770 |

(f) **Loss function.** Using the full loss term greatly contributes to the best results. $\mathcal{L}_{\text{rec}}$ loss is crucial to training an effective spike-to-image model.

|  | PSNR | SSIM | LPIPS |
|---|---|---|---|
| w/o $\mathcal{L}_{\text{rec}}$ | 36.43 | 0.957 | 0.0227 |
| w/o $\mathcal{L}_{\text{per}}$ | 37.30 | 0.954 | 0.0446 |
| w/o $\mathcal{L}_{\text{msc}}$ | 38.54 | 0.964 | 0.0194 |
| Total loss | **38.79** | **0.966** | **0.0183** |

for energy-efficient spike-to-image reconstruction. Note that, except for SSIR, the other comparison methods adopt CNNs. 3) **Event-based**: ET-Net [48] and HyperE2VID [10], where our proposed HSER is cascaded with the classical event-to-image reconstruction architectures. 4) **CNN-based**: Spk2ImgNet [55] and WGSE [52], which are SOTA spike-to-image reconstruction methods.

## 5.2 Comparison with SOTA Methods

As shown in Table 1, our approach significantly outperforms SOTA methods in terms of reconstruction accuracy on both synthetic and real datasets. Apart from SNN-based SSIR [57], which has limited performance despite being computationally efficient, our method enjoys the lowest model complexity and competitive model size. CNN-based spike-to-image methods incur high inference costs due to their step-by-step paradigm. Notably, on average, our model is $11\times$ faster than Spk2ImgNet [55] and $5\times$ faster than WGSE [52]. Moreover, event-based architectures have limited adaptability. Traditional methods show unsatisfactory reconstruction quality due to restricted modeling power.

The qualitative results are presented in Fig. 4. We can see that our method produces perceptually more pleasing and higher-fidelity images, especially for the edges of fast-moving objects. For instance, in cases involving pedestrians, drones, and small balls, our method achieves sharper edges, less noise, and fewer blurring and aliasing artifacts. Note that our method also ensures fast reconstruction of intensity frames (*cf*., Fig. 2), which further highlights its potential for practical applications.

## 5.3 Ablation Studies

To verify the effectiveness of the proposed method, we conduct a series of ablation studies from the perspective of network architecture and loss function on the SREDS dataset [57].

**Ablation on Spike Embedding Representation.** We implement various spike embedding representation methods, including explicit, implicit, and combined. The multidilated representation [49] stacks multiple dilated convolutions for a larger receptive field, while the hierarchical spatial-temporal (HiST) representation [58] integrates multi-scale 3D convolutions for feature fusion, both of which have been applied for optical flow estimation. As shown in Table 2, despite

Table 2: Ablation on spike embedding representation.

| Explicit | Implicit | | | PSNR↑ | SSIM↑ | LPIPS↓ |
|---|---|---|---|---|---|---|
| TFP | Multi-dilated | HiST | ResNet | | | |
| ✓ |  |  |  | 36.39 | 0.951 | 0.0260 |
|  | ✓ |  |  | 36.71 | 0.953 | 0.0305 |
|  |  | ✓ |  | 37.95 | 0.960 | 0.0212 |
|  |  |  | ✓ | 38.01 | 0.962 | 0.0201 |
| ✓ | ✓ |  |  | 37.60 | 0.959 | 0.0225 |
| ✓ |  | ✓ |  | 38.06 | 0.962 | 0.0206 |
| ✓ |  |  | ✓ | **38.79** | **0.966** | **0.0183** |

being straightforward, ResNet [21] proves to be a relatively more workable spike representation method. Our HSER organically combines TFP and ResNet, which takes full advantage of both explicit and implicit representations, thus achieving the best result. Note that, even using only the simplest TFP [65], our method demonstrates competitive performance, which also demonstrates the effectiveness of our spatio-temporal interactive learning architecture.

**Ablation on Feature Pyramid Level.** We investigate the influence of varying hierarchical features. As shown in Table 3a, even with just a 3-level pyramid, our method significantly outperforms existing CNN-based methods in terms of model size and computational efficiency, while still guaranteeing a leading performance. As the pyramid level increases, it will introduce finer-grained spatio-temporal interaction, which is conducive to achieving better image reconstruction quality.

**Ablation on Motion-Intensity Interactive Block.** As reported in Table 3b, removing the warping-based inter-frame feature alignment results in sub-optimal performance, indicating that temporal contextual information is beneficial for intermediate frame recovery. Notably, the overall performance is severely degraded when the synthesis-based intra-feature filtering is removed, demonstrating the essential role of intermediate features in reconstructing the target frame. When the motion-intensity interaction is performed simultaneously from coarse to fine, a superior performance is obtained.

**Ablation on Symmetric Interactive Attention Block.** We either feed the top-level pyramid features directly into the decoder or use a standard cross-attention beforehand that independently models unilateral feature correlations. Due to the symmetric bilateral feature interaction, which facilitates more accurate context awareness, our method achieves superior performance, as shown in Table 3c. Also, our interactive attention has small parameters and FLOPs, ensuring lightweight network design.

**Ablation on Multi-Motion Field Estimation Block.** We propose estimating multiple motion fields at the bottom-level pyramid to compensate for more contextual details. As shown in Table 3d, using multiple groups of motion fields yields higher reconstruction quality, consistent with [30, 24]. As the number of motion fields increases, the model exhibits minor performance fluctuations. Still, it achieves gains over models based on a single group of motion fields.

**Ablation on Model Capacity.** We apply a width multiplier [22] to the feature channels based on the current configuration. Table 3e presents that increasing the model capacity has a positive effect, indicating that our architecture is highly flexible. Particularly, the parameter size and computational cost of $\times 0.5$ are ahead of SOTA methods, and it also has commendable reconstruction capability.

**Ablation on Loss Function.** We evaluate the impact of different loss terms in Table 3f. It is evident that our total loss function is effective, as it performs the best when all loss terms are included.

# 6 Conclusion

In this paper, we proposed an efficient spike-to-image reconstruction method based on spatio-temporal interactive learning. In particular, a joint motion-intensity learning architecture was designed to perform inter-frame feature alignment and intra-frame feature filtering progressively. Moreover, we introduced a symmetric interactive attention block and a multi-motion field estimation block for bilateral correlation modeling and context detail compensation. Extensive experiments on synthetic and real data have demonstrated that our approach has excellent performance in reconstruction quality and inference speed, while also enjoying good flexibility and applicability.

**Limitations.** Our proposed HSER, similar to other spike embedding representation methods [56, 52, 60, 59, 58, 49], implicitly assumes that the scene has sufficient illumination, such that the image reconstruction can be achieved based on spike streams with a fixed temporal length. However, in extremely low-light scenarios, the limited accumulated light intensity within a fixed temporal length results in darker reconstructed images and increased noise, adversely affecting the visual experience. Note that this is a common problem for current spike-to-image reconstruction methods [56, 52, 65, 63, 57]. We plan to extend our model to handle these issues in future work.

**Acknowledgments.** This work was supported by National Science and Technology Major Project (2021ZD0109803), Beijing Natural Science Foundation (L233024), Beijing Municipal Science & Technology Commission, Administrative Commission of Zhongguancun Science Park (Z241100003524012), and National Natural Science Foundation of China (62088102, 62136001, 62276007, 62401021). Bin Fan was also supported by China National Postdoctoral Program for Innovative Talents (BX20230013) and China Postdoctoral Science Foundation (2024M750101).

# References

[1] Christian Brandli, Raphael Berner, Minhao Yang, Shih-Chii Liu, and Tobi Delbruck. A 240× 180 130 db 3 $\mu$s latency global shutter spatiotemporal vision sensor. *IEEE Journal of Solid-State Circuits (JSSC)*, 49(10):2333–2341, 2014.

[2] Pablo Rodrigo Gantier Cadena, Yeqiang Qian, Chunxiang Wang, and Ming Yang. SPADE-E2VID: Spatially-adaptive denormalization for event-based video reconstruction. *IEEE Transactions on Image Processing (TIP)*, 30:2488–2500, 2021.

[3] Yakun Chang, Yeliduosi Xiaokaiti, Yujia Liu, Bin Fan, Zhaojun Huang, Tiejun Huang, and Boxin Shi. Towards HDR and HFR video from rolling-mixed-bit spikings. In *Proceedings of the IEEE Conference on Computer Vision and Pattern Recognition (CVPR)*, 2024.

[4] Shiyan Chen, Chaoteng Duan, Zhaofei Yu, Ruiqin Xiong, and Tiejun Huang. Self-supervised mutual learning for dynamic scene reconstruction of spiking camera. In *Proceedings of the International Joint Conferences on Artificial Intelligence (IJCAI)*, page 2859–2866, 2022.

[5] Shiyan Chen, Zhaofei Yu, and Tiejun Huang. Self-supervised joint dynamic scene reconstruction and optical flow estimation for spiking camera. In *Proceedings of the AAAI Conference on Artificial Intelligence (AAAI)*, pages 350–358, 2023.

[6] Shiyan Chen, Jiyuan Zhang, Yajing Zheng, Tiejun Huang, and Zhaofei Yu. Enhancing motion deblurring in high-speed scenes with spike streams. *Proceedings of the Advances in Neural Information Processing Systems (NeurIPS)*, 36, 2023.

[7] Siwei Dong, Tiejun Huang, and Yonghong Tian. Spike camera and its coding methods. In *Proceedings of the Data Compression Conference (DCC)*, pages 437–437, 2017.

[8] Yanchen Dong, Ruiqin Xiong, Jing Zhao, Jian Zhang, Xiaopeng Fan, Shuyuan Zhu, and Tiejun Huang. Joint demosaicing and denoising for spike camera. In *Proceedings of the AAAI Conference on Artificial Intelligence (AAAI)*, pages 1582–1590, 2024.

[9] Yanchen Dong, Jing Zhao, Ruiqin Xiong, and Tiejun Huang. High-speed scene reconstruction from low-light spike streams. In *Proceedings of the IEEE International Conference on Visual Communications and Image Processing (VCIP)*, pages 1–5. IEEE, 2022.

[10] Burak Ercan, Onur Eker, Canberk Saglam, Aykut Erdem, and Erkut Erdem. HyperE2VID: Improving event-based video reconstruction via hypernetworks. *IEEE Transactions on Image Processing (TIP)*, 2024.

[11] Bin Fan and Yuchao Dai. Inverting a rolling shutter camera: Bring rolling shutter images to high framerate global shutter video. In *Proceedings of the IEEE International Conference on Computer Vision (ICCV)*, pages 4228–4237, 2021.

[12] Bin Fan, Yuchao Dai, and Mingyi He. SUNet: Symmetric undistortion network for rolling shutter correction. In *Proceedings of the IEEE International Conference on Computer Vision (ICCV)*, pages 4541–4550, 2021.

[13] Bin Fan, Yuchao Dai, and Mingyi He. Rolling shutter camera: Modeling, optimization and learning. *Machine Intelligence Research*, 20(6):783–798, 2023.

[14] Bin Fan, Yuchao Dai, and Hongdong Li. Rolling shutter inversion: Bring rolling shutter images to high framerate global shutter video. *IEEE Transactions on Pattern Analysis and Machine Intelligence*, 45(5):6214–6230, 2022.

[15] Bin Fan, Yuchao Dai, and Hongdong Li. Learning bilateral cost volume for rolling shutter temporal super-resolution. *IEEE Transactions on Pattern Analysis and Machine Intelligence*, 46(5):3862–3879, 2024.

[16] Bin Fan, Yuchao Dai, Zhiyuan Zhang, Qi Liu, and Mingyi He. Context-aware video reconstruction for rolling shutter cameras. In *Proceedings of the IEEE Conference on Computer Vision and Pattern Recognition (CVPR)*, pages 17572–17582, 2022.

[17] Bin Fan, Ying Guo, Yuchao Dai, Chao Xu, and Boxin Shi. Self-supervised learning for rolling shutter temporal super-resolution. *IEEE Transactions on Circuits and Systems for Video Technology*, 2024.

[18] Bin Fan, Yuxin Mao, Yuchao Dai, Zhexiong Wan, and Qi Liu. Joint appearance and motion learning for efficient rolling shutter correction. In *Proceedings of the IEEE Conference on Computer Vision and Pattern Recognition (CVPR)*, pages 5671–5681, 2023.

[19] Guillermo Gallego, Tobi Delbrück, Garrick Orchard, Chiara Bartolozzi, Brian Taba, Andrea Censi, Stefan Leutenegger, Andrew J Davison, Jörg Conradt, Kostas Daniilidis, et al. Event-based vision: A survey. *IEEE Transactions on Pattern Analysis and Machine Intelligence (T-PAMI)*, 44(1):154–180, 2020.

[20] Kaiming He, Xiangyu Zhang, Shaoqing Ren, and Jian Sun. Delving deep into rectifiers: Surpassing human-level performance on imagenet classification. In *Proceedings of the IEEE International Conference on Computer Vision (ICCV)*, pages 1026–1034, 2015.

[21] Kaiming He, Xiangyu Zhang, Shaoqing Ren, and Jian Sun. Deep residual learning for image recognition. In *Proceedings of the IEEE Conference on Computer Vision and Pattern Recognition (CVPR)*, pages 770–778, 2016.

[22] Andrew G Howard, Menglong Zhu, Bo Chen, Dmitry Kalenichenko, Weijun Wang, Tobias Weyand, Marco Andreetto, and Hartwig Adam. MobileNets: Efficient convolutional neural networks for mobile vision applications. *arXiv preprint arXiv:1704.04861*, 2017.

[23] Liwen Hu, Rui Zhao, Ziluo Ding, Lei Ma, Boxin Shi, Ruiqin Xiong, and Tiejun Huang. Optical flow estimation for spiking camera. In *Proceedings of the IEEE Conference on Computer Vision and Pattern Recognition (CVPR)*, pages 17844–17853, 2022.

[24] Ping Hu, Simon Niklaus, Stan Sclaroff, and Kate Saenko. Many-to-many splatting for efficient video frame interpolation. In *Proceedings of the IEEE Conference on Computer Vision and Pattern Recognition (CVPR)*, pages 3553–3562, 2022.

[25] Gao Huang, Zhuang Liu, Laurens Van Der Maaten, and Kilian Q Weinberger. Densely connected convolutional networks. In *Proceedings of the IEEE Conference on Computer Vision and Pattern Recognition (CVPR)*, pages 4700–4708, 2017.

[26] Tiejun Huang, Yajing Zheng, Zhaofei Yu, Rui Chen, Yuan Li, Ruiqin Xiong, Lei Ma, Junwei Zhao, Siwei Dong, Lin Zhu, Jianing Li, Shanshan Jia, Yihua Fu, Boxin Shi, Si Wu, and Yonghong Tian. $1000\times$ faster camera and machine vision with ordinary devices. *Engineering*, 25:110–119, 2022.

[27] Taewoo Kim, Yujeong Chae, Hyun-Kurl Jang, and Kuk-Jin Yoon. Event-based video frame interpolation with cross-modal asymmetric bidirectional motion fields. In *Proceedings of the IEEE Conference on Computer Vision and Pattern Recognition (CVPR)*, pages 18032–18042, 2023.

[28] Diederik P Kingma and Jimmy Ba. Adam: A method for stochastic optimization. In *Proceedings of the International Conference on Learning Representations (ICLR)*, 2015.

[29] Yu Li, Ye Zhu, Ruoteng Li, Xintao Wang, Yue Luo, and Ying Shan. Hybrid warping fusion for video frame interpolation. *International Journal of Computer Vision (IJCV)*, 130(12):2980–2993, 2022.

[30] Zhen Li, Zuo-Liang Zhu, Ling-Hao Han, Qibin Hou, Chun-Le Guo, and Ming-Ming Cheng. AMT: All-pairs multi-field transforms for efficient frame interpolation. In *Proceedings of the IEEE Conference on Computer Vision and Pattern Recognition (CVPR)*, pages 9801–9810, 2023.

[31] Patrick Lichtsteiner, Christoph Posch, and Tobi Delbruck. A $128\times128$ 120 db $15\mu s$ latency asynchronous temporal contrast vision sensor. *IEEE Journal of Solid-State Circuits (JSSC)*, 43(2):566–576, 2008.

[32] Ze Liu, Yutong Lin, Yue Cao, Han Hu, Yixuan Wei, Zheng Zhang, Stephen Lin, and Baining Guo. Swin transformer: Hierarchical vision transformer using shifted windows. In *Proceedings of the IEEE International Conference on Computer Vision (ICCV)*, pages 10012–10022, 2021.

[33] Liying Lu, Ruizheng Wu, Huaijia Lin, Jiangbo Lu, and Jiaya Jia. Video frame interpolation with transformer. In *Proceedings of the IEEE Conference on Computer Vision and Pattern Recognition (CVPR)*, pages 3532–3542, 2022.

[34] Richard H Masland. The neuronal organization of the retina. *Neuron*, 76(2):266–280, 2012.

[35] Anish Mittal, Anush Krishna Moorthy, and Alan Conrad Bovik. No-reference image quality assessment in the spatial domain. *IEEE Transactions on Image Processing (TIP)*, 21(12):4695–4708, 2012.

[36] Anish Mittal, Rajiv Soundararajan, and Alan C Bovik. Making a "completely blind" image quality analyzer. *IEEE Signal Processing Letters (SPL)*, 20(3):209–212, 2012.

[37] Diederik Paul Moeys, Federico Corradi, Chenghan Li, Simeon A Bamford, Luca Longinotti, Fabian F Voigt, Stewart Berry, Gemma Taverni, Fritjof Helmchen, and Tobi Delbruck. A sensitive dynamic and active pixel vision sensor for color or neural imaging applications. *IEEE Transactions on Biomedical Circuits and Systems*, 12(1):123–136, 2017.

[38] Seungjun Nah, Sungyong Baik, Seokil Hong, Gyeongsik Moon, Sanghyun Son, Radu Timofte, and Kyoung Mu Lee. NTIRE 2019 challenge on video deblurring and super-resolution: Dataset and study. In *Proceedings of the IEEE Conference on Computer Vision and Pattern Recognition Workshops (CVPRW)*, pages 1996–2005, 2019.

[39] Junheum Park, Jintae Kim, and Chang-Su Kim. BiFormer: Learning bilateral motion estimation via bilateral transformer for 4K video frame interpolation. In *Proceedings of the IEEE Conference on Computer Vision and Pattern Recognition (CVPR)*, pages 1568–1577, 2023.

[40] Christoph Posch, Daniel Matolin, and Rainer Wohlgenannt. A QVGA 143 db dynamic range frame-free PWM image sensor with lossless pixel-level video compression and time-domain CDS. *IEEE Journal of Solid-State Circuits (JSSC)*, 46(1):259–275, 2010.

[41] Henri Rebecq, René Ranftl, Vladlen Koltun, and Davide Scaramuzza. Events-to-video: Bringing modern computer vision to event cameras. In *Proceedings of the IEEE Conference on Computer Vision and Pattern Recognition (CVPR)*, pages 3857–3866, 2019.

[42] Cedric Scheerlinck, Henri Rebecq, Daniel Gehrig, Nick Barnes, Robert Mahony, and Davide Scaramuzza. Fast image reconstruction with an event camera. In *Proceedings of the Winter Conference on Applications of Computer Vision (WACV)*, pages 156–163, 2020.

[43] Timo Stoffregen, Cedric Scheerlinck, Davide Scaramuzza, Tom Drummond, Nick Barnes, Lindsay Kleeman, and Robert Mahony. Reducing the sim-to-real gap for event cameras. In *Proceedings of the European Conference on Computer Vision (ECCV)*, pages 534–549. Springer, 2020.

[44] Stepan Tulyakov, Alfredo Bochicchio, Daniel Gehrig, Stamatios Georgoulis, Yuanyou Li, and Davide Scaramuzza. Time Lens++: Event-based frame interpolation with parametric non-linear flow and multi-scale fusion. In *Proceedings of the IEEE Conference on Computer Vision and Pattern Recognition (CVPR)*, pages 17755–17764, 2022.

[45] Stepan Tulyakov, Daniel Gehrig, Stamatios Georgoulis, Julius Erbach, Mathias Gehrig, Yuanyou Li, and Davide Scaramuzza. Time Lens: Event-based video frame interpolation. In *Proceedings of the IEEE Conference on Computer Vision and Pattern Recognition (CVPR)*, pages 16155–16164, 2021.

[46] Yixuan Wang, Jianing Li, Lin Zhu, Xijie Xiang, Tiejun Huang, and Yonghong Tian. Learning stereo depth estimation with bio-inspired spike cameras. In *Proceedings of the IEEE International Conference on Multimedia & Expo (ICME)*, pages 1–6. IEEE, 2022.

[47] Heinz Wässle. Parallel processing in the mammalian retina. *Nature Reviews Neuroscience*, 5(10):747–757, 2004.

[48] Wenming Weng, Yueyi Zhang, and Zhiwei Xiong. Event-based video reconstruction using transformer. In *Proceedings of the IEEE International Conference on Computer Vision (ICCV)*, pages 2563–2572, 2021.

[49] Lujie Xia, Ziluo Ding, Rui Zhao, Jiyuan Zhang, Lei Ma, Zhaofei Yu, Tiejun Huang, and Ruiqin Xiong. Unsupervised optical flow estimation with dynamic timing representation for spike camera. *Proceedings of the Advances in Neural Information Processing Systems (NeurIPS)*, 36, 2023.

[50] Sidi Yang, Tianhe Wu, Shuwei Shi, Shanshan Lao, Yuan Gong, Mingdeng Cao, Jiahao Wang, and Yujiu Yang. MANIQA: Multi-dimension attention network for no-reference image quality assessment. In *Proceedings of the IEEE Conference on Computer Vision and Pattern Recognition (CVPR)*, pages 1191–1200, 2022.

[51] Syed Waqas Zamir, Aditya Arora, Salman Khan, Munawar Hayat, Fahad Shahbaz Khan, and Ming-Hsuan Yang. Restormer: Efficient transformer for high-resolution image restoration. In *Proceedings of the IEEE Conference on Computer Vision and Pattern Recognition (CVPR)*, pages 5728–5739, 2022.

[52] Jiyuan Zhang, Shanshan Jia, Zhaofei Yu, and Tiejun Huang. Learning temporal-ordered representation for spike streams based on discrete wavelet transforms. In *Proceedings of the AAAI Conference on Artificial Intelligence (AAAI)*, pages 137–147, 2023.

[53] Jiyuan Zhang, Lulu Tang, Zhaofei Yu, Jiwen Lu, and Tiejun Huang. Spike transformer: Monocular depth estimation for spiking camera. In *Proceedings of the European Conference on Computer Vision (ECCV)*, pages 34–52. Springer, 2022.

[54] Richard Zhang, Phillip Isola, Alexei A Efros, Eli Shechtman, and Oliver Wang. The unreasonable effectiveness of deep features as a perceptual metric. In *Proceedings of the IEEE Conference on Computer Vision and Pattern Recognition (CVPR)*, pages 586–595, 2018.

[55] Jing Zhao, Ruiqin Xiong, Hangfan Liu, Jian Zhang, and Tiejun Huang. Spk2ImgNet: Learning to reconstruct dynamic scene from continuous spike stream. In *Proceedings of the IEEE Conference on Computer Vision and Pattern Recognition (CVPR)*, pages 11996–12005, 2021.

[56] Jing Zhao, Ruiqin Xiong, Jiyu Xie, Boxin Shi, Zhaofei Yu, Wen Gao, and Tiejun Huang. Reconstructing clear image for high-speed motion scene with a retina-inspired spike camera. *IEEE Transactions on Computational Imaging (TCI)*, 8:12–27, 2021.

[57] Rui Zhao, Ruiqin Xiong, Jian Zhang, Zhaofei Yu, Shuyuan Zhu, Lei Ma, and Tiejun Huang. Spike camera image reconstruction using deep spiking neural networks. *IEEE Transactions on Circuits and Systems for Video Technology (TCSVT)*, 2023.

[58] Rui Zhao, Ruiqin Xiong, Jian Zhang, Xinfeng Zhang, Zhaofei Yu, and Tiejun Huang. Optical flow for spike camera with hierarchical spatial-temporal spike fusion. In *Proceedings of the AAAI Conference on Artificial Intelligence (AAAI)*, pages 7496–7504, 2024.

[59] Rui Zhao, Ruiqin Xiong, Jing Zhao, Zhaofei Yu, Xiaopeng Fan, and Tiejun Huang. Learning optical flow from continuous spike streams. *Proceedings of the Advances in Neural Information Processing Systems (NeurIPS)*, 35:7905–7920, 2022.

[60] Rui Zhao, Ruiqin Xiong, Jing Zhao, Jian Zhang, Xiaopeng Fan, Zhaofei Yu, and Tiejun Huang. Boosting spike camera image reconstruction from a perspective of dealing with spike fluctuations. In *Proceedings of the IEEE Conference on Computer Vision and Pattern Recognition (CVPR)*, 2024.

[61] Yajing Zheng, Jiyuan Zhang, Rui Zhao, Jianhao Ding, Shiyan Chen, Ruiqin Xiong, Zhaofei Yu, and Tiejun Huang. SpikeCV: Open a continuous computer vision era. *arXiv preprint arXiv:2303.11684*, 2023.

[62] Yajing Zheng, Lingxiao Zheng, Zhaofei Yu, Tiejun Huang, and Song Wang. Capture the moment: High-speed imaging with spiking cameras through short-term plasticity. *IEEE Transactions on Pattern Analysis and Machine Intelligence (T-PAMI)*, 45(7):8127–8142, 2023.

[63] Yajing Zheng, Lingxiao Zheng, Zhaofei Yu, Boxin Shi, Yonghong Tian, and Tiejun Huang. High-speed image reconstruction through short-term plasticity for spiking cameras. In *Proceedings of the IEEE Conference on Computer Vision and Pattern Recognition (CVPR)*, pages 6358–6367, 2021.

[64] Lin Zhu, Xianzhang Chen, Xiao Wang, and Hua Huang. Finding visual saliency in continuous spike stream. In *Proceedings of the AAAI Conference on Artificial Intelligence (AAAI)*, pages 7757–7765, 2024.

[65] Lin Zhu, Siwei Dong, Tiejun Huang, and Yonghong Tian. A retina-inspired sampling method for visual texture reconstruction. In *Proceedings of the IEEE International Conference on Multimedia & Expo (ICME)*, pages 1432–1437, 2019.

[66] Lin Zhu, Siwei Dong, Jianing Li, Tiejun Huang, and Yonghong Tian. Retina-like visual image reconstruction via spiking neural model. In *Proceedings of the IEEE Conference on Computer Vision and Pattern Recognition (CVPR)*, pages 1438–1446, 2020.

# A  More Visual Results

We additionally present more spike-to-image reconstruction results. Figs. 5, 6, and 8 show qualitative comparisons on "momVidarReal2021" [61], "SREDS" [57], and "recVidarReal2019" [66] datasets, respectively. It can be observed that compared to the baseline methods, our approach consistently and robustly produces the most satisfactory image reconstruction results, with fewer aliasing artifacts and clearer object outlines, *e.g.*, the intricate structures of distant buildings, the tight and dense keycaps of a keyboard, and the steel cables and railings of the bridge shot from a high-speed train (350km/h), *etc*. In particular, the image reconstruction results on our real-captured spike data are exemplified in Fig. 9. One can see that our approach also effectively recovers finer and more accurate image details with less noise, providing a better visual experience. These experiments also fully validate the excellent generalization ability of our proposed method.

# B  Further Analysis of Hybrid Spike Embedding Representation

In Table 2 of the main manuscript, we have demonstrated the simplicity and effectiveness of the proposed hybrid spike embedding representation (HSER). Here, we further validate the superiority of HSER by integrating different spike embedding representations into HyperE2VID [10] (*i.e.*, replacing the event voxel grid with our HSER). Note that HyperE2VID [10] is a recently proposed SOTA event-to-image reconstruction method. As reported in Table 4, using the explicit TFP [65] alone yields a seemingly feasible result. Notably, in the CNN-based implicit spike embedding representations (*e.g.*, [49, 58]), ResNet [21] remains a more effective strategy even as a regular tool. In contrast, when the vanilla TFP is organically combined with ResNet (*i.e.*, our HSER), the best reconstruction accuracy is achieved. This is mainly because our HSER efficiently merges the certainty of explicit representation with the strong expressive power of implicit representation. Note that our HSER also ensures faster inference speed for the overall network. Additionally, our HSER can better adapt the event-to-image reconstruction model to spike-to-image reconstruction, which is a good indication that the key reconstruction module is transferable by adjusting the frontmost embedding representation. In conclusion, our HSER is a concise and effective spike embedding representation paradigm, which can be used to further enhance the performance of off-the-shelf image reconstruction methods.

# C  Visualization of Ablation Results

In Table 3 of the main manuscript, we have quantitatively evaluated the effectiveness of the proposed network architecture and loss function. We recognize that the primary innovation of our joint learning architecture lies in the interactive and joint perspective to handle temporal and spatial information simultaneously. It is this holistic approach that sets our method apart and yields superior performance, as opposed to depending solely on high-performing individual components. We also understand that our model is handy to scale to fit into diverse scenarios. For instance, in scenarios that demand high precision and have abundant computational resources, a larger model is preferred. Conversely, for mobile or real-time applications, a simpler model can be employed.

Here, we further visually present more ablation results based on the "recVidarReal2019" dataset [66], as shown in Fig. 7. It can be seen that removing the warping-based inter-frame feature alignment as well as estimating only a single group of motion fields at the bottom-level pyramid hinders the complementary exploitation of temporal contextual information, especially for fast-moving objects, resulting in unpleasant visual artifacts. Note that removing the synthetic-based intra-frame feature filtering leads to catastrophic failure of the model. Furthermore, removing the symmetric interactive attention block impedes the construction of bilateral correlations, which adversely affects the subsequent joint motion-intensity refinement. Lastly, removing the reconstruction loss $\mathcal{L}_{\text{rec}}$ is also detrimental to the reconstruction of high-quality images. In contrast, thanks to our joint learning architecture design, our full model can promote more efficient spatio-temporal interaction, thereby reconstructing higher-fidelity intermediate images.

# D  Effectiveness Validation of High Frame Rate Video Reconstruction

When inputting a continuous spike stream, our approach is capable of reconstructing consecutive video frames sequentially. In Fig. 10, we further illustrate the high frame rate video reconstruction

Table 4: Quantitative comparisons of HyperE2VID [10] under different spike embedding representations on the SREDS dataset [57]. The runtime is tested using a single RTX 3090 GPU on real-captured data [66] with a spatial resolution of $400 \times 250$, similar to Fig. 2 in the main manuscript.

| Explicit | Implicit | | | PSNR↑ | SSIM↑ | LPIPS↓ | Runtime (ms) |
|---|---|---|---|---|---|---|---|
| TFP [65] | Multi-dilated [49] | HiST [58] | ResNet [21] | | | | |
| ✓ | | | | 31.48 | 0.866 | 0.1523 | 6.14 |
| | ✓ | | | 35.62 | 0.940 | 0.0611 | 7.56 |
| | | ✓ | | 34.92 | 0.938 | 0.0616 | 14.6 |
| | | | ✓ | 36.18 | 0.946 | 0.0520 | 6.47 |
| ✓ | ✓ | | | 35.58 | 0.937 | 0.0692 | 8.19 |
| ✓ | | ✓ | | 35.19 | 0.940 | 0.0599 | 14.9 |
| ✓ | | | ✓ | **36.37** | **0.947** | **0.0506** | 6.84 |

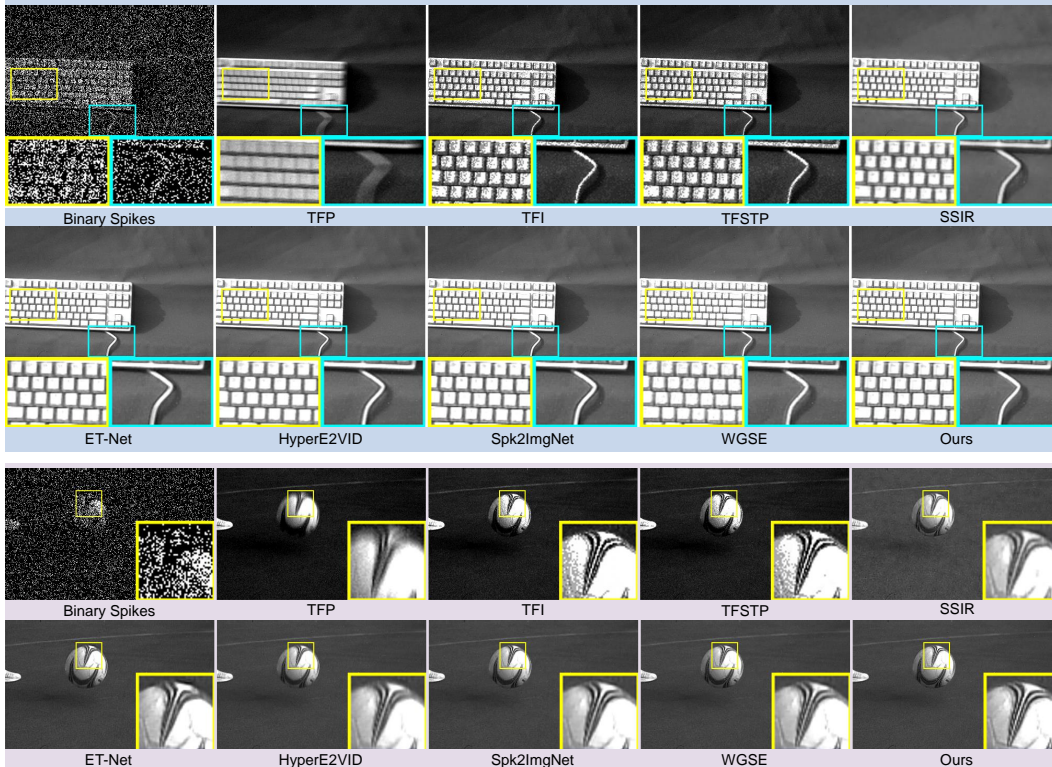

Figure 5: More qualitative comparison on the real-captured "momVidarReal2021" dataset [61]. Our reconstructed images exhibit sharper and clearer edge detail on objects like keyboards and footballs. Please zoom in for more details.

results based on our real-collected spike data, including filming a rapidly spinning fan as well as recording the instantaneous process when a water balloon bursts. It is evident that our approach can generate smooth and consistent consecutive image sequences faithfully and accurately, in which rich image content is restored, such as fan leaves and water splashes, demonstrating its effectiveness in practical applications, such as capturing remarkable high-speed motion moments.

# E   Broader Impact

The method proposed herein provides an efficient solution for dynamic scene reconstruction by leveraging continuous spike streams with high temporal resolution. Our proposed technique may hold potential benefits for a variety of real-world applications and users, especially in scenarios involving high-speed motion. As the goal of spike-to-image reconstruction is to reproduce real scene details, our method may not pose negative ethical implications if we do not discuss specific scene content.

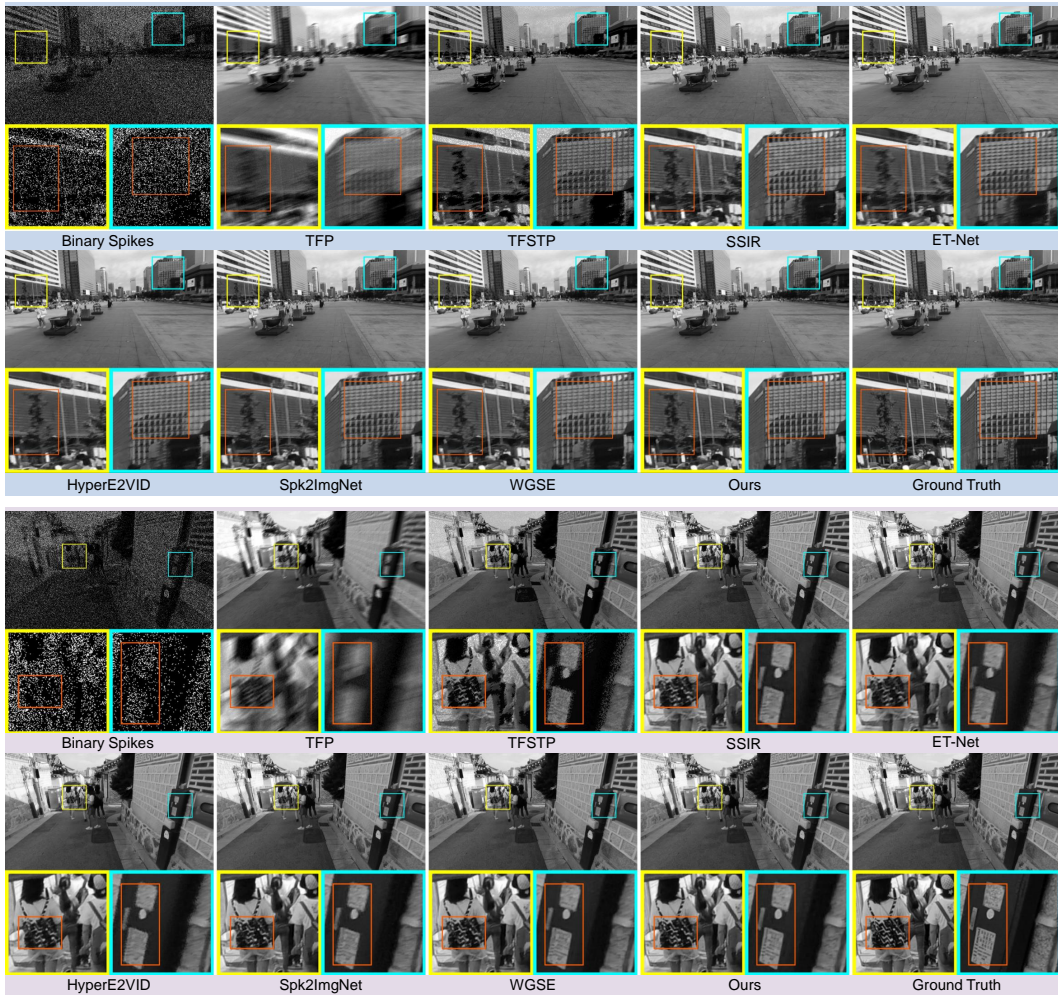

Figure 6: More qualitative comparison on the synthetic "SREDS" dataset [57]. Our method shows excellent reconstruction results for both the complex structures of distant buildings and the backpacks of nearby pedestrians. Please zoom in for more details.

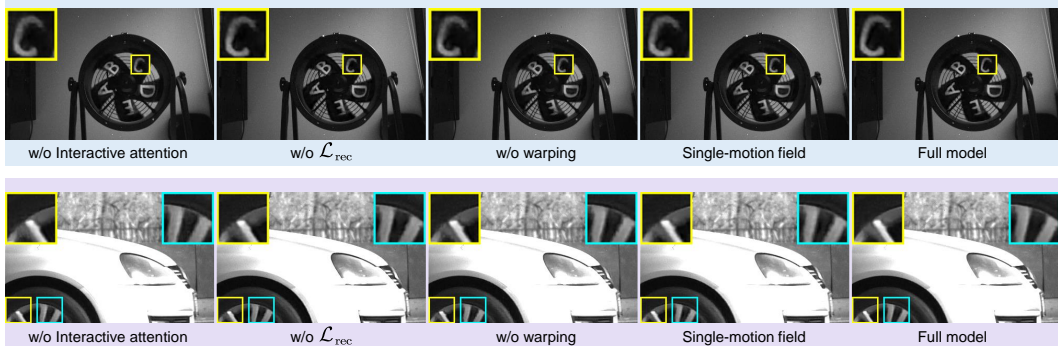

Figure 7: Visualization of ablation results based on the "recVidarReal2019" dataset [66]. From left to right, we show the removal of symmetric interactive attention block, the removal of reconstruction loss $\mathcal{L}_{\mathrm{rec}}$, the removal of warping-based inter-frame feature alignment, and the estimation of single-motion field. Our full model reconstructs higher-fidelity images with fewer artifacts. Please zoom in for more details.

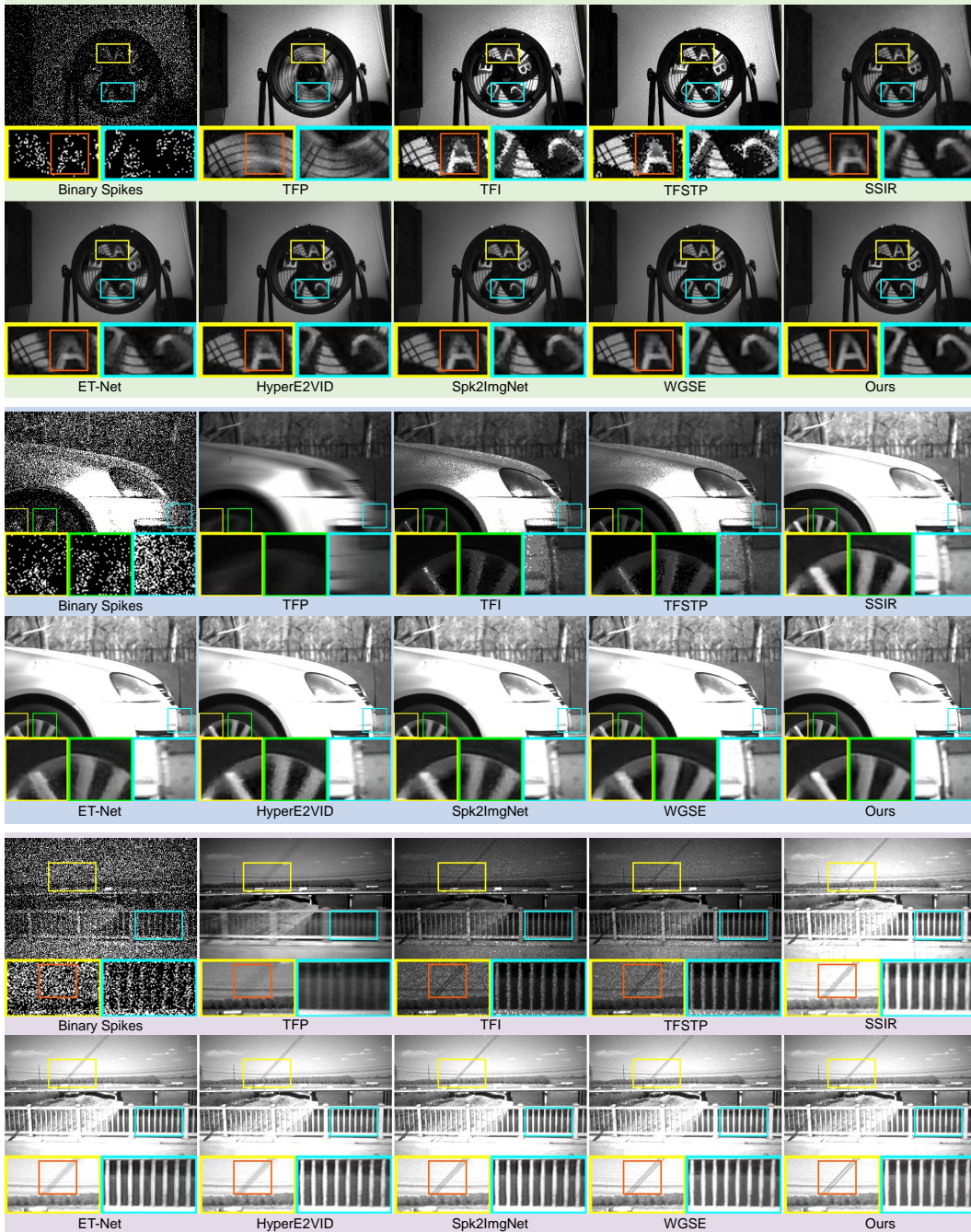

Figure 8: Qualitative comparison on the real-captured "recVidarReal2019" dataset [66]. Whether photographing a high-speed rotating fan (2600 rpm) and a fast-moving car (100 km/h), or shooting from a high-speed train (350 km/h), our method recovers more image details and more accurate structure. Please zoom in for more details.

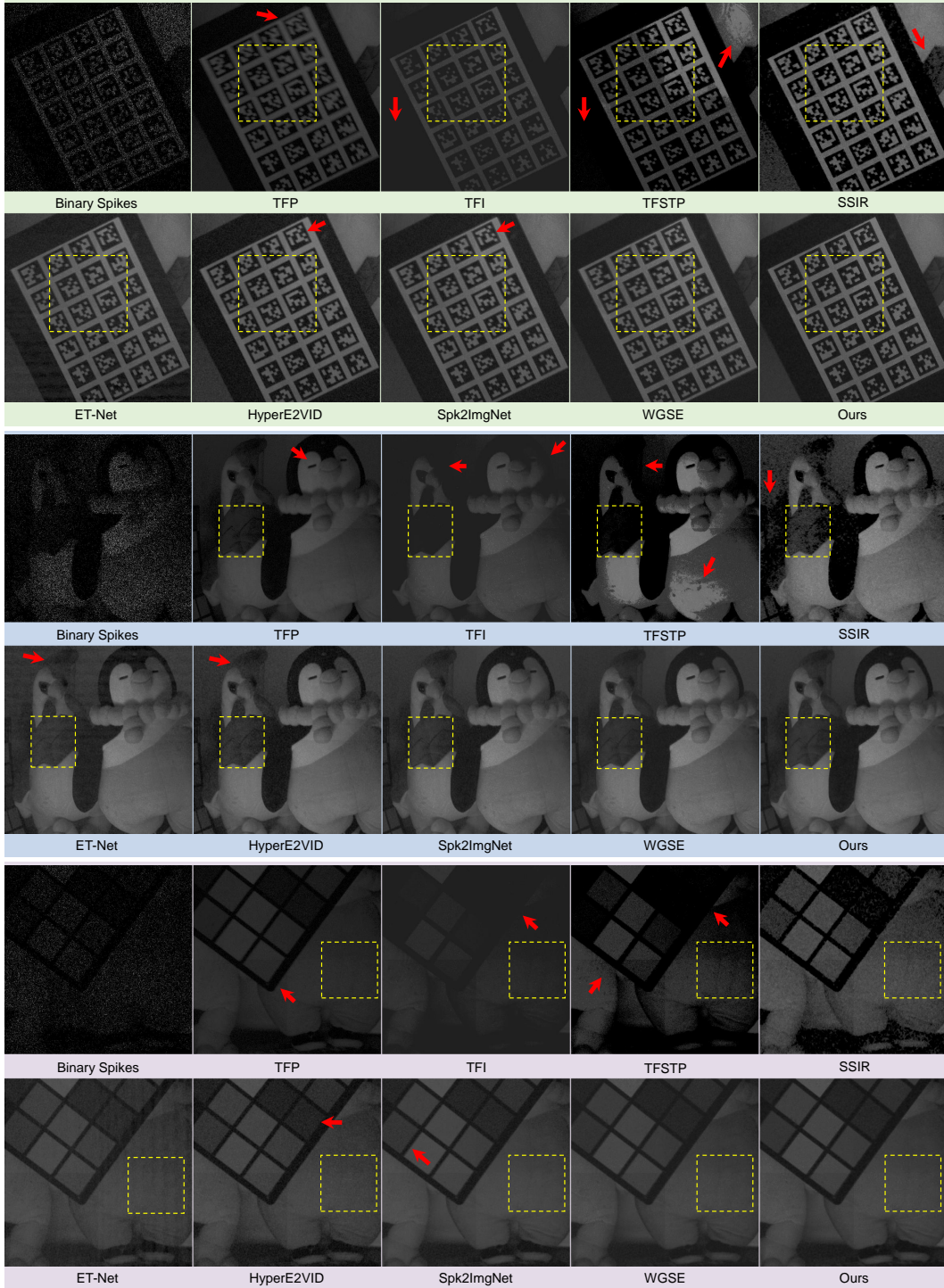

Figure 9: Qualitative comparison on our real-captured spike data. Our method can suppress noise and restore more accurate details more efficiently overall. Yellow dashed boxes and red arrows indicate these regions. Please zoom in for more details.

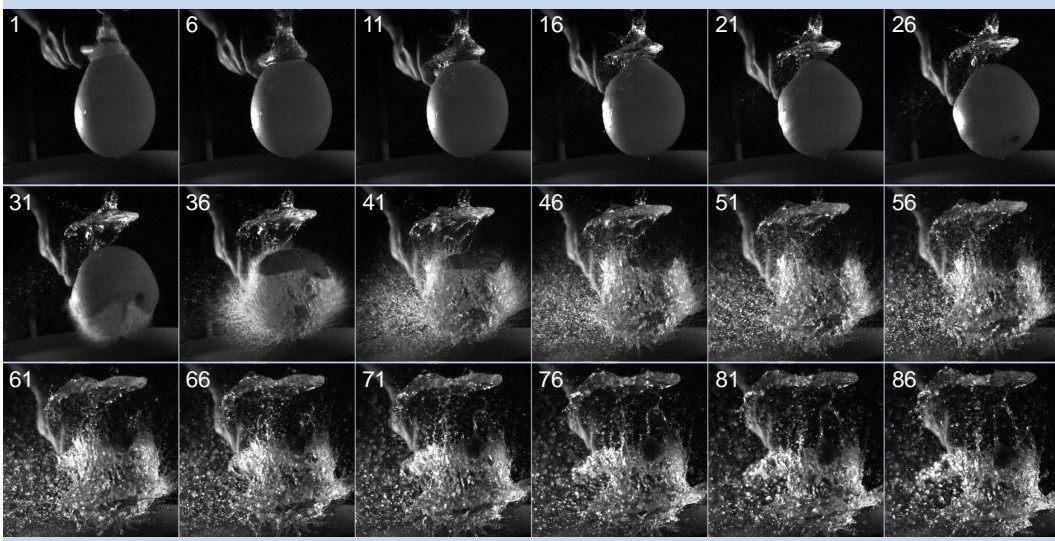

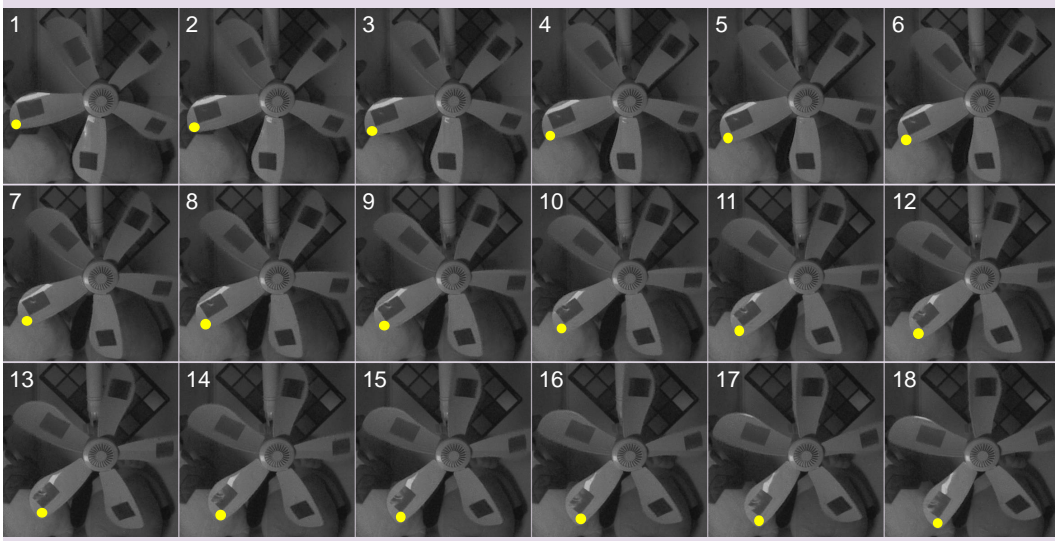

Figure 10: High frame rate video reconstruction results on real spike data we captured with a spiking camera. The temporal sequence of 18 intensity frames is visualized, including two high-speed scenes, *i.e.*, a bursting water balloon and a rapidly spinning fan (∼750 rpm). We add a yellow dot to indicate the rotation of the fan leaves.

